# Dynamic Bayesian Networks for Brain-Computer Interfaces

**Pradeep Shenoy**
Department of Computer Science
University of Washington
Seattle, WA 98195
pshenoy@cs.washington.edu

**Rajesh P. N. Rao**
Department of Computer Science
University of Washington
Seattle, WA 98195
rao@cs.washington.edu

## Abstract

We describe an approach to building brain-computer interfaces (BCI) based on graphical models for probabilistic inference and learning. We show how a dynamic Bayesian network (DBN) can be used to infer probability distributions over brain- and body-states during planning and execution of actions. The DBN is learned directly from observed data and allows measured signals such as EEG and EMG to be interpreted in terms of internal states such as intent to move, preparatory activity, and movement execution. Unlike traditional classification-based approaches to BCI, the proposed approach (1) allows continuous tracking and prediction of internal states over time, and (2) generates control signals based on an entire probability distribution over states rather than binary yes/no decisions. We present preliminary results of brain- and body-state estimation using simultaneous EEG and EMG signals recorded during a self-paced left/right hand movement task.

## 1   Introduction

The problem of building a brain-computer interface (BCI) has received considerable attention in recent years. Several researchers have demonstrated the feasibility of using EEG signals as a non-invasive medium for building human BCIs [1, 2, 3, 4, 5] (see also [6] and articles in the same issue). A central theme in much of this research is the postulation of a discrete *brain state* that the user maintains while performing one of a set of physical or imagined actions. The goal is to decode the hidden brain state from the observable EEG signal, and to use the decoded state to control a robot or a cursor on a computer screen.

Most previous approaches to BCI (e.g., [1, 2, 4]) have utilized classification methods applied to time slices of EEG data to discriminate between a small set of brain states (e.g., left versus right hand movement). These methods typically involve various forms of preprocessing (such as band-pass filtering or temporal smoothing) as well as feature extraction on time slices known to contain one of the chosen set of brain states. The output of the classifier is typically a yes/no decision regarding class membership. A significant drawback of such an approach is the need to have a "point of reference" for the EEG data, i.e., a synchronization point in time where the behavior of interest was performed. Also, classifier-based approaches typically do not model the uncertainty in their class estimates. As a result, it

is difficult to have a continuous estimate of the brain state and to associate an uncertainty with the current estimate.

In this paper, we propose a new framework for BCI based on probabilistic graphical models [7] that overcomes some of the limitations of classification-based approaches to BCI. We model the dynamics of hidden brain- and body-states using a Dynamic Bayesian Network (DBN) that is learned directly from EEG and EMG data. We show how a DBN can be used to infer probability distributions over hidden state variables, where the state variables correspond to brain states useful for BCI (such as "Intention to move left hand", "Left hand in motion", etc). Using a DBN gives us several advantages in addition to providing a continuous probabilistic estimate of brain state. First, it allows us to explicitly model the hidden causal structure and dependencies between different brain states. Second, it facilitates the integration of information from multiple modalities such as EEG and EMG signals, allowing, for example, EEG-derived estimates to be bootstrapped from EMG-derived estimates. In addition, learning a dynamic graphical model for time-varying data such as EEG allows other useful operations such as prediction, filling in of missing data, and smoothing of state estimates using information from future data points. These capabilities are difficult to obtain while working exclusively in the frequency domain or using whole slices of the data (or its features) for training classifiers. We illustrate our approach in a simple Left versus Right hand movement task and present preliminary results showing supervised learning and Bayesian inference of hidden state for a dataset containing simultaneous EEG and EMG recordings.

## 2   The DBN Framework

We study the problem of modeling spontaneous movement of the left/right arm using EEG and EMG signals. It is well known that EEG signals show a slow potential drift prior to spontaneous motor activity. This potential drift, known as the Bereitschaftspotential (BP, see [8] for an excellent survey), shows variation in distribution over scalp with respect to the body part being moved. In particular, the BP related to movement of left versus right arm shows a strong lateral asymmetry. This allows one to not only estimate the intent to move *prior to* actual movement, but also distinguish between left and right movements. Previous approaches [1, 2] have utilized BP signals in classification-based BCI protocols based on synchronization cues that identify points of movement onset. In our case, the challenge was to model the structure of BPs and related movement signals using the states of the DBN, and to recognize actions without explicit synchronization cues.

Figure 1 shows the complete DBN (referred to as $N_{full}$ in this paper) used to model the left-right hand movement task. The hidden state $B_t$ in Figure 1(a) tracks the higher-level brain state over time and generates the hidden EEG and EMG states $E_t$ and $M_t$ respectively. These hidden states in turn generate the observed EEG and EMG signals. The dashed arrows indicate that the hidden states make transitions over time. As shown in Figure 1(b), the state $B_t$ is intended to model the high-level intention of the subject. The figure shows both the values $B_t$ can take as well the constraints on the transition between values. The actual probabilities of the allowed transitions are learned from data.

The hidden states $E_t$ and $M_t$ are intended to model the temporal structure of the EEG and EMG signals, which are generated using a mixture of Gaussians conditioned on $E_t$ and $M_t$ respectively. In the same way as the values of $B_t$ are customized for our particular experiment, we would like the state transitions of $E_t$ and $M_t$ to also reflect their respective constraints. This is important since it allows us to independently learn the simpler DBN $N_{emg}$ consisting of only the node $M_t$ and the observed EMG signal. Similarly, we can also independently learn the model $N_{eeg}$ consisting of the node $E_t$ and the observed EEG signal. We use the models shown in Figure 2 for allowed transitions of the states $M_t$ and $E_t$ respectively. In particular, Figure 2(a) indicates that the EMG state can transition along one

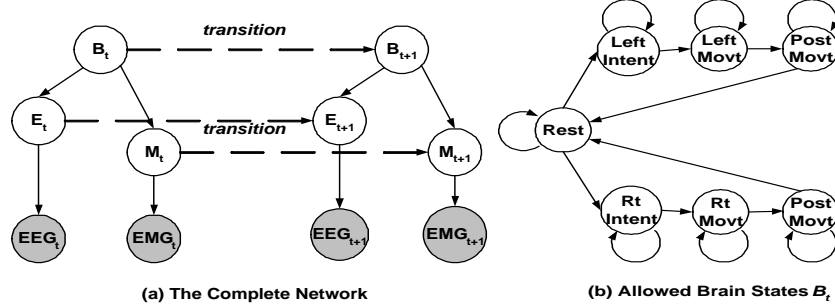

**(a) The Complete Network**    **(b) Allowed Brain States $B_t$**

Figure 1: **Dynamic graphical model for modeling brain and body processes in a self-paced movement task**: (a) At each time instant $t$, the brain state $B_t$ generates the EEG and EMG internal states $E_t$ and $M_t$ respectively, which in turn generate the observed EEG and EMG. The dotted arrows represent transitions to a state at the next time step. (b) The transition graph for the brain state $B_t$. The probability of each allowed transition is learned from input data.

of three *chains of states* (labeled (1), (2), and (3)), representing the rest state, a left-hand action and a right-hand action respectively. In each chain, the state $M_t$ in each time step either retains its old value with a given probability (self-pointing arrow) or transitions to the next state value in that particular chain. The transition graph of Figure 2(b) shows similar constraints on the EEG, except that the left and right action chains are further partitioned into intent, action, and post-action subgroups of states, since each of these components are discernible from the BP in EEG (but not from EMG) signals.

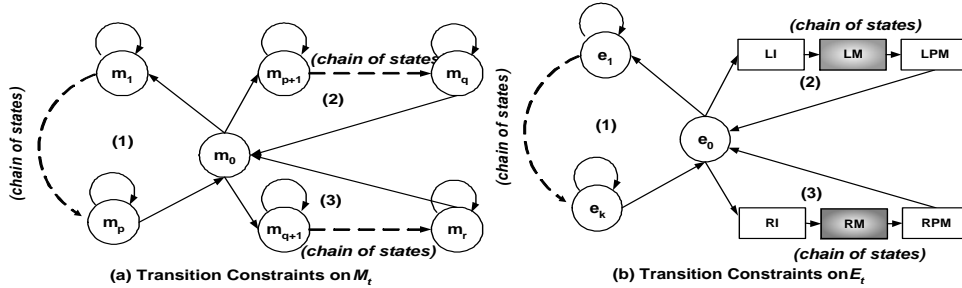

**(a) Transition Constraints on $M_t$**    **(b) Transition Constraints on $E_t$**

Figure 2: **Constrained transition graphs for the hidden EMG and EEG states $E_t$ and $M_t$ respectively**. (a) The EMG state transitions between its values $m_i$ are constrained to be in one of three chains: the chains model (1) rest, (2) left arm movement, and (3) right arm movement. (b) In the EEG state transition graph, the left and right movement chains are further divided into state values encoding intent (LI/RI), movement (LM/RM), and post movement (LPM/RPM).

# 3 Experiments and Results

## 3.1 Data Collection and Processing

**The task:** The subject pressed two distinct keys on a keyboard with the left hand or right

hand at random at a self-initiated pace. We recorded 8 EEG channels around the motor area of cortex (C3, Cz, C4, FC1, FC2, CP1, CP2, Pz) using averaged ear electrodes as reference, and 2 differential pairs of EMG (one on each arm). Data was recorded at 2048Hz for a period of 20 minutes, with the movements being separated by approximately 3-4s.

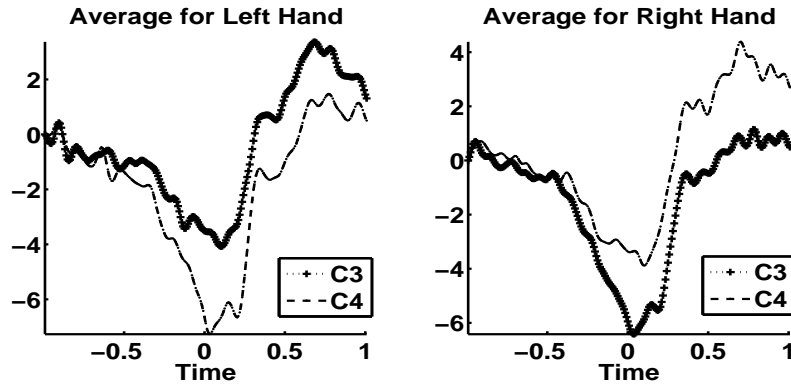

Figure 3: **Movement-related potential drift recorded during the hand-movement task**: The two plots show the EEG signals averaged over all trials from the motor-related channels C3 and C4 for left (left panel) and right hand movement (right panel). The averages indicate the onset and laterality of upcoming movements.

**Processing:** The EEG channels were bandpass-filtered 0.5Hz-5Hz, before being downsampled and smoothed at 128Hz. The EMG channels were converted to RMS values computed over windows for an effective sampling rate of 128Hz.

**Data Analysis:** The recorded data were first analyzed in the traditional manner by averaging across all trials. Figure 3 shows the average of EEG channels C3 and C4 for left and right hand movement actions respectively. As can be seen, the averages for both channels are different for the two classes. Furthermore, there is a slow potential drift *preceding* the action and a return to the baseline potential after the action is performed. Previous researchers [1] have classified EEG data over a window leading up to the instant of action with high accuracy (over 90%) into left or right movement classes. Thus, there appears to be a reliable amount of information in the EEG signal for at least discriminating between left versus right movements.

**Data Evaluation using SVMs:** To obtain a baseline and to evaluate the quality of our recorded data, we tested the performance of linear support vector machines (SVMs) on classifying our EEG data into left and right movement classes. The choice of linear SVMs was motivated by their successful use on similar problems by other researchers [1]. Time slices of 0.5 seconds before each movement were concatenated from all EEG channels and used for classification. We performed hyper-parameter selection using leave-one-out cross-validation on 15 minutes of data and obtained an error of 15% on the remaining 5 minutes of data. Such an error rate is comparable to those obtained in previous studies on similar tasks, suggesting that the recorded data contains sufficient movement-related information to be tested in experiments involving DBNs.

**Learning the parameters of the DBN:** We used the Graphical Models Toolkit (GMTK) [9] for learning the parameters of our DBN. GMTK provides support for expressing constraints on state transitions (as described in Section 2). It learns the constrained conditional probability tables and the parameters for the mixture of Gaussians using the expectation-maximization (EM) algorithm.

We constructed a supervisory signal from the recorded key-presses as follows: A period of

100ms around each keystroke was labeled "motor action" for the appropriate hand. This signal was used to train the network $N_{emg}$ in a supervised manner. To generate a supervisory signal for the network $N_{eeg}$, or the full combined network $N_{full}$ (Figure 1), we added prefixes and postfixes of 150ms each to each action in this signal, and labeled them "preparatory" and "post-movement" activity respectively. These time-periods were chosen by examining the average EEG and EMG activity over all actions. Thus, we can use partial (EEG only) or full evidence in the inference step to obtain probability distributions over brain state. The following sections describe our learning procedure and inference results in greater detail.

### 3.2   Learning and Inference with EMG

Our first step is to learn the simpler model $N_{emg}$ that has only the hidden $M_t$ state and the observed EMG signal. This is to test inference using the EMG signal alone. The parameters of this DBN were learned in a supervised manner.

We used 15 minutes of EMG data to train our simplified model, and then tested it on the remaining 5 minutes of data. The model was tested using Viterbi decoding (a single pass of max-product inference over the network). In other words, the maximum a posteriori (MAP) sequence of values for hidden states was computed. Figure 4 shows a 100s slice of data containing 2 channels of EMG, and the predicted hidden EMG state $M_t$. The states 0, 1 and 2 correspond to "no action", left, and right actions respectively. In the shown figure, the state $M_t$ successfully captures not only all the obvious arm movements but also the actions that are obscured by noise.

### 3.3   Learning the EEG Model

We used the supervisory signal described earlier to learn the corresponding EEG model $N_{eeg}$. Note that the brain-state can be inferred from the hidden EEG state $E_t$ directly, since the state space is appropriately partitioned as shown in Figure 2(b).

Figure 5 shows the result of inference on the learned model $N_{eeg}$ using only the EEG signals as evidence. The figure shows a subset of the EEG channels (C3,Cz,C4), the supervisory signal, and the predicted brain state $B_t$ (the MAP estimate). The figure shows that many of the instances of action (but not all) are correctly identified by the model.

Our model gives us at each time instant a MAP-estimated state sequence that best describes the past, and the probability associated with that state sequence. This gives us, at each time instant, a measure of how likely each brain state $B_t$ is, with reference to the others. For convenience, we can use the probability associated with the REST state (see Figure 1) as reference. Figure 6 shows a graphical illustration of this instantaneous time estimate. The plotted graphs are, in order, the supervisory signal (i.e., the "ground truth value") and the instantaneous measures of likelihood of intention/movement/post-movement states for the left and right hand respectively. For convenience, we represent the likelihood ratio of each state's MAP probability estimate to that of the rest state, and use a logarithmic scale. We see that the true hand movements are correctly inferred in a surprisingly large number of cases (log likelihood ratio crosses 0). Furthermore, the actual likelihood values convey a measure of the uncertainty in the inference, a property that would be of great value for critical BCI applications such as controlling a robotic wheelchair.

In summary, our graphical models $N_{emg}$ and $N_{eeg}$ have shown promising results in correctly identifying movement onset from EMG and EEG signals respectively. Ongoing work is focused on improving accuracy by using features extracted from EEG, and inference using both EEG and EMG in $N_{full}$ (the full model).

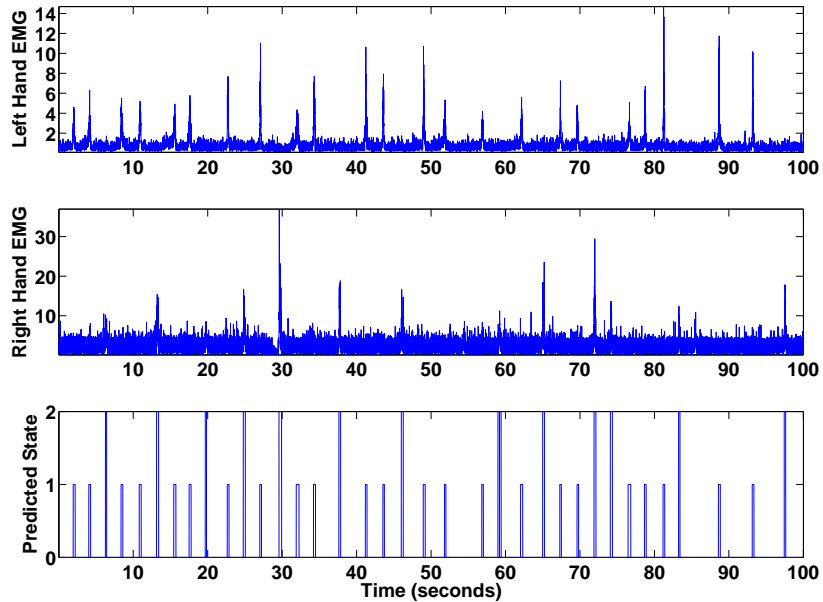

Figure 4: **Bayesian Inference of Movement using EMG**: The figure shows 100 seconds of EMG data from two channels along with the MAP state sequence predicted by our trained EMG model. The states 0,1,2 correspond to "no action", left, and right actions respectively. Our model correctly identifies the obscured spikes in the noisy right EMG channel

## 4   Discussion and Conclusion

We have shown that dynamic Bayesian networks (DBNs) can be used to model the transitions between brain- and muscle-states as a subject performs a motor task. In particular, a two-level hierarchical network was proposed for simultaneously estimating higher-level brain state and lower-level EEG and EMG states in a left/right hand movement task. The results demonstrate that for a self-paced movement task, hidden brain states useful for BCI such as intention to move the left or right hand can be decoded from a DBN learned directly from EEG and EMG data.

Previous work on BCIs can be grouped into two broad classes: self-regulatory BCIs and BCIs based on detecting brain state. Self-regulatory BCIs rely on training the user to regulate certain features of the EEG, such as cortical positivity [10], or oscillatory activity (the $\mu$ rhythm, see [5]), in order to control, for example, a cursor on a display. The approach presented in this paper falls in the second class of BCIs, those based on detecting brain states [1, 2, 3, 4]. However, rather than employing classification methods, we use probabilistic graphical models for inferring brain state and learning the transition probabilities between brain states.

Successfully learning a dynamic graphical model as suggested in this paper offers several advantages over traditional classification-based schemes for BCI. It allows one to explicitly model the hidden causal structure and dependencies between different brain states. It provides a probabilistic framework for integrating information from multiple modalities

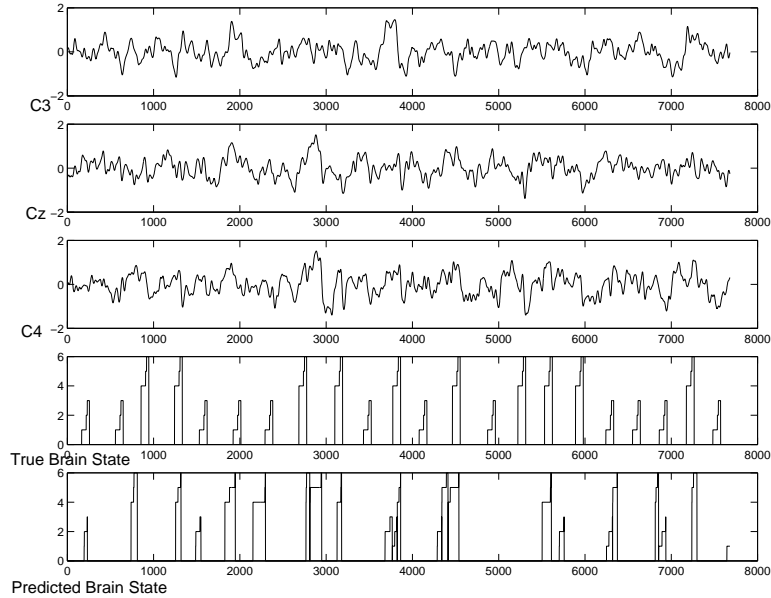

Figure 5: **Bayesian Inference of Brain State using EEG**: The figure shows 1 minute of EEG data (at 128Hz) for the channels C3, Cz, C4, along with the "true" brain state and the brain state inferred using our DBN model with only EEG evidence. State 0 is the rest state, states 1 through 3 represent left hand movement, and 4 through 6 represent right hand movement (see Figure 1(b)).

such as EEG and EMG signals, allowing, for example, EEG-derived estimates to be bootstrapped from EMG-derived estimates. A dynamic graphical model for time-varying data such as EEG also allows prediction, filling in of missing data, and smoothing of state estimates using information from future data points, properties not easily achieved in methods that work exclusively in the frequency domain or use data slices for training classifiers. Our current efforts are focused on investigating methods for learning dynamic graphical models for motor tasks of varying complexity and using these models to build robust, probabilistic BCI systems.

# References

[1] B. Blankertz, G. Curio, and K.R. Mueller. Classifying single trial EEG: Towards brain computer interfacing. In *Advances in Neural Information Processing Systems 12*, 2001.

[2] G. Dornhege, B. Blankertz, G. Curio, and K.-R. Mueller. Combining features for BCI. In *Advances in Neural Information Processing Systems 15*, 2003.

[3] J. D. Bayliss and D. H. Ballard. Recognizing evoked potentials in a virtual environment. In *Advances in Neural Information Processing Systems 12*, 2000.

[4] P. Meinicke, M. Kaper, F. Hoppe, M. Heumann, and H. Ritter. Improving transfer rates in brain computer interfacing: a case study. In *Advances in Neural Information Processing Systems 15*, 2003.

[5] J.R. Wolpaw, D.J. McFarland, and T.M. Vaughan. Brain-computer interfaces for communication and control. *IEEE Trans Rehab Engg*, pages 222–226, 2000.

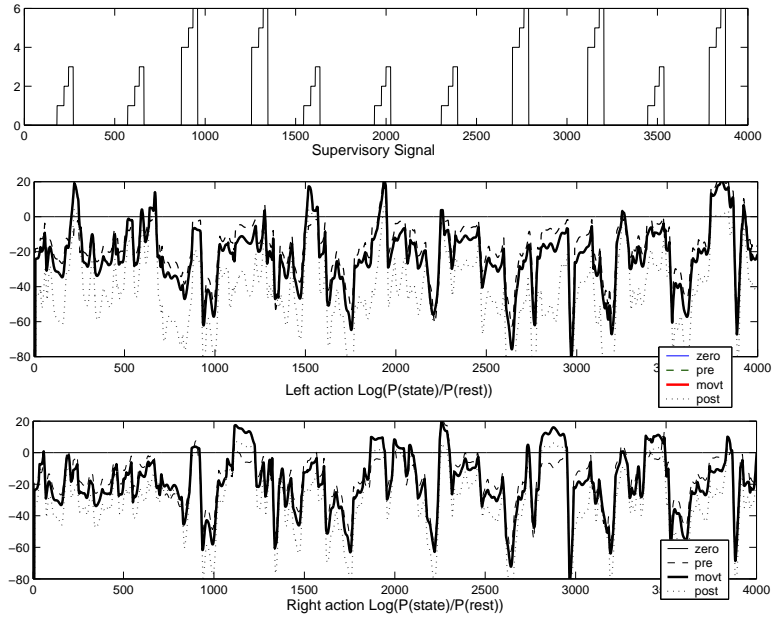

Figure 6: **Probabilistic Estimation of Brain State**: The figure shows the supervisory signal, along with a probabilistic measure of the current state for left and right actions respectively. The measure shown is the log ratio of the instantaneous MAP estimate for the relevant state and the estimate for the rest state.

[6] J. R. Wolpaw et al. Brain-computer interface technology: a review of the first international meeting. *IEEE Trans Rehab Engg*, 8:164–173, 2000.

[7] R. E. Neapolitan. *Learning Bayesian Networks*. Prentice Hall, NJ, 2004.

[8] M. Jahanshahi and M. Hallet. *The Bereitschaftspotential: movement related cortical potentials*. Kluwer Academic, New York, 2002.

[9] J. Bilmes and G. Zweig. The graphical models toolkit: An open source software system for speech and time-series processing. In *IEEE Intl. Conf. on Acoustics, Speech, and Signal Processing, Orlando FL*, 2002.

[10] N. Birbaumer, N. Ghanayim, T. Hinterberger, I. Iverson, B. Kotchubey, A. Kiibler, J. Perelmouter, E. Taub, and H. Flor. A spelling device for the paralyzed. In *Nature, 398: 297-298*, 1999.
